# Robust Kernel Principal Component Analysis

**Minh Hoai Nguyen** & **Fernando De la Torre**
Carnegie Mellon University, Pittsburgh, PA 15213, USA.

## Abstract

Kernel Principal Component Analysis (KPCA) is a popular generalization of linear PCA that allows non-linear feature extraction. In KPCA, data in the input space is mapped to higher (usually) dimensional feature space where the data can be linearly modeled. The feature space is typically induced implicitly by a kernel function, and linear PCA in the feature space is performed via the kernel trick. However, due to the implicitness of the feature space, some extensions of PCA such as robust PCA cannot be directly generalized to KPCA. This paper presents a technique to overcome this problem, and extends it to a unified framework for treating noise, missing data, and outliers in KPCA. Our method is based on a novel cost function to perform inference in KPCA. Extensive experiments, in both synthetic and real data, show that our algorithm outperforms existing methods.

## 1 Introduction

Principal Component Analysis (PCA) [9] is one of the primary statistical techniques for feature extraction and data modeling. One drawback of PCA is its limited ability to model non-linear structures that exist in many computing applications. Kernel methods [18] enable us to extend PCA to model non-linearities while retaining its computational efficiency. In particular, Kernel PCA (KPCA) [19] has repeatedly outperformed PCA in many image modeling tasks [19, 14].

Unfortunately, realistic visual data is often corrupted by undesirable artifacts due to occlusion (*e.g.* a hand in front of a face, Fig. 1.d), illumination (e.g. specular refection, Fig. 1.e), noise (e.g. from capturing device, Fig.1.b), or from the underlying data generation method (e.g. missing data due to transmission, Fig. 1.c). Therefore, robustness to noise, missing data, and outliers is a desired property to have for algorithms in computer vision.

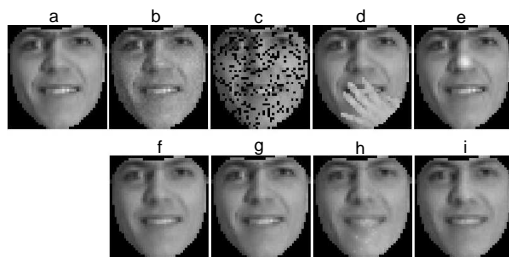

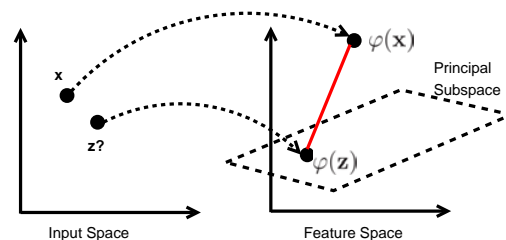

Figure 1: Several types of data corruption and results of our method. a) original image, b) corruption by additive Gaussian noise, c) missing data, d) hand occlusion, e) specular reflection. f) to i) are the results of our method for recovering uncorrupted data from b) to e) respectively.

Figure 2: Using KPCA principal subspace to find **z**, a clean version of corrupted sample **x**.

Throughout the years, several extensions of PCA have been proposed to address the problems of outliers and missing data, see [6] for a review. However, it still remains unclear how to generalize those extensions to KPCA; since directly migrating robust PCA techniques to KPCA is not possible

due to the implicitness of the feature space. To overcome this problem, in this paper, we propose Robust KPCA (RKPCA), a unified framework for denoising images, recovering missing data, and handling intra-sample outliers. Robust computation in RKPCA does not suffer from the implicitness of the feature space because of a novel cost function for reconstructing "clean" images from corrupted data. The proposed cost function is composed of two terms, requiring the reconstructed image to be close to the KPCA principal subspace as well as to the input sample. We show that robustness can be naturally achieved by using robust functions to measure the closeness between the reconstructed and the input data.

## 2 Previous work

### 2.1 KPCA and pre-image

KPCA [19, 18, 20] is a non-linear extension of principal component analysis (PCA) using kernel methods. The kernel represents an implicit mapping of the data to a (usually) higher dimensional space where linear PCA is performed.

Let $\mathcal{X}$ denote the input space and $\mathcal{H}$ the feature space. The mapping function $\varphi : \mathcal{X} \to \mathcal{H}$ is implicitly induced by a kernel function $k : \mathcal{X} \times \mathcal{X} \to \Re$ that defines the similarity between data in the input space. One can show that if $k(\cdot, \cdot)$ is a kernel then the function $\varphi(\cdot)$ and the feature space $\mathcal{H}$ exist; furthermore $k(\mathbf{x}, \mathbf{y}) = \langle \varphi(\mathbf{x}), \varphi(\mathbf{y}) \rangle$ [18].

However, directly performing linear PCA in the feature space might not be feasible because the feature space typically has very high dimensionality (including infinity). Thus KPCA is often done via the kernel trick. Let $\mathbf{D} = [\mathbf{d}_1 \ \mathbf{d}_2 \ ... \ \mathbf{d}_n]$, see notation[1], be a training data matrix, such that $\mathbf{d}_i \in \mathcal{X} \ \forall i = \overline{1, n}$. Let $k(\cdot, \cdot)$ denote a kernel function, and $\mathbf{K}$ denote the kernel matrix (element $ij$ of $\mathbf{K}$ is $k_{ij} = k(\mathbf{d}_i, \mathbf{d}_j)$). KPCA is computed in closed form by finding first $m$ eigenvectors ($\mathbf{a}_i$'s) corresponding to the largest eigenvalues ($\lambda_i$'s) of the kernel matrix $\mathbf{K}$ (i.e. $\mathbf{KA} = \mathbf{A\Lambda}$). The eigenvectors in the feature space $\mathbf{V}$ can be computed as $\mathbf{V} = \mathbf{\Gamma A}$, where $\mathbf{\Gamma} = [\varphi(\mathbf{d}_1)...\varphi(\mathbf{d}_n)]$. To ensure orthonormality of $\{\mathbf{v}_i\}_{i=1}^m$, KPCA imposes that $\lambda_i \langle \mathbf{a}_i, \mathbf{a}_i \rangle = 1$. It can be shown that $\{\mathbf{v}_i\}_{i=1}^m$ form an orthonormal basis of size $m$ that best preserves the variance of data in the feature space [19].

Assume $\mathbf{x}$ is a data point in the input space, and let $\mathbf{P}\varphi(\mathbf{x})$ denote the projection of $\varphi(\mathbf{x})$ onto the principal subspace $\{\mathbf{v}_i\}_1^m$. Because $\{\mathbf{v}_i\}_1^m$ is a set of orthonormal vectors, we have $\mathbf{P}\varphi(\mathbf{x}) = \sum_{i=1}^m \langle \varphi(\mathbf{x}), \mathbf{v}_i \rangle \mathbf{v}_i$. The reconstruction error (in feature space) is given by:

$$E_{proj}(\mathbf{x}) = ||\varphi(\mathbf{x}) - \mathbf{P}\varphi(\mathbf{x})||_2^2 = \langle \varphi(\mathbf{x}), \varphi(\mathbf{x}) \rangle - \sum \langle \varphi(\mathbf{x}), \mathbf{v}_i \rangle^2 = k(\mathbf{x}, \mathbf{x}) - r(\mathbf{x})^T \mathbf{M} r(\mathbf{x}),$$

$$\text{where } r(\mathbf{x}) = \mathbf{\Gamma}^T \varphi(\mathbf{x}), \text{ and } \mathbf{M} = \sum \mathbf{a}_i \mathbf{a}_i^T . \tag{1}$$

The pre-image of the projection is the $\mathbf{z} \in \mathcal{X}$ that satisfies $\varphi(\mathbf{z}) = \mathbf{P}\varphi(\mathbf{x})$; $\mathbf{z}$ is also referred to as the KPCA reconstruction of $\mathbf{x}$. However, the pre-image of $\mathbf{P}\varphi(\mathbf{x})$ usually does not exist, so finding the KPCA reconstruction of $\mathbf{x}$ means finding $\mathbf{z}$ such that $\varphi(\mathbf{z})$ is as close to $\mathbf{P}\varphi(\mathbf{x})$ as possible. It should be noted that the closeness between $\varphi(\mathbf{z})$ and $\mathbf{P}\varphi(\mathbf{x})$ can be defined in many ways, and different cost functions lead to different optimization problems. Schölkopf *et al* [17] and Mika *et al* [13] propose to approximate the reconstruction of $\mathbf{x}$ by $\arg\min_{\mathbf{z}} ||\varphi(\mathbf{z}) - \mathbf{P}\varphi(\mathbf{x})||_2^2$. Two other objective functions have been proposed by Kwok & Tsang [10] and Bakir *et al* [2].

### 2.2 KPCA-based algorithms for dealing with noise, outliers and missing data

Over the years, several methods extending KPCA algorithms to deal with noise, outliers, or missing data have been proposed. Mika *et al* [13], Kwok & Tsang [10], and Bakir *et al* [2] show how denoising can be achieved by using the pre-image. While these papers present promising denoising results for handwritten digits, there are at least two problems with these approaches. Firstly, because the input image $\mathbf{x}$ is noisy, the similarity measurement between $\mathbf{x}$ and other data point $\mathbf{d}_i$ (i.e. $k(\mathbf{x}, \mathbf{d}_i)$ the kernel) might be adversely affected, biasing the KPCA reconstruction of $\mathbf{x}$. Secondly,

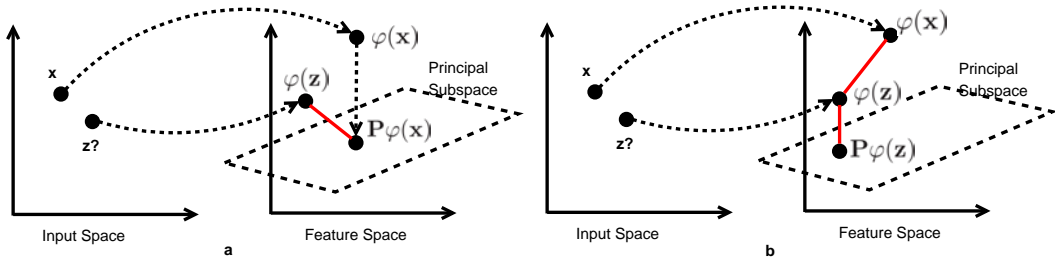

Figure 3: Key difference between previous work (a) and ours (b). In (a), one seeks $\mathbf{z}$ such that $\varphi(\mathbf{z})$ is close to $\mathbf{P}\varphi(\mathbf{x})$. In (b), we seek $\mathbf{z}$ such that $\varphi(\mathbf{z})$ is close to both $\varphi(\mathbf{x})$ and the principal subspace.

current KPCA reconstruction methods equally weigh all the features (i.e. pixels); it is impossible to weigh the importance of some features over the others.

Other existing methods also have limitations. Some [7, 22, 1] only consider robustness of the principal subspace; they do not address robust fitting. Lu *et al* [12] present an iterative approach to handle outliers in training data. At each iteration, the KPCA model is built, and the data points that have the highest reconstruction errors are regarded as outliers and discarded from the training set. However, this approach does not handle intra-sample outliers (outliers that occur at a pixel level [6]). Several other approaches also considering Berar *et al* [3] propose to use KPCA with polynomial kernels to handle missing data. However, it is not clear how to extend this approach to other kernels. Furthermore, with polynomial kernels of high degree, the objective function is hard to optimize. Sanguinetti & Lawrence [16] propose an elegant framework to handle missing data. The framework is based on the probabilistic interpretation inherited from Probabilistic PCA [15, 21, 11]. However, Sanguinetti & Lawrence [16] do not address the problem of outliers.

This paper presents a novel cost function that unifies the treatment of noise, missing data and outliers in KPCA. Experiments show that our algorithm outperforms existing approaches [6, 10, 13, 16].

## 3 Robust KPCA

### 3.1 KPCA reconstruction revisited

Given an image $\mathbf{x} \in \mathcal{X}$, Fig. 2 describes the task of finding the KPCA-reconstructed image of $\mathbf{x}$ (uncorrupted version of $\mathbf{x}$ to which we will refer as KPCA reconstruction). Mathematically, the task is to find a point $\mathbf{z} \in \mathcal{X}$ such that $\varphi(\mathbf{z})$ is in the principal subspace (denote $\mathcal{PS}$) and $\varphi(\mathbf{z})$ is as close to $\varphi(\mathbf{x})$ as possible. In other words, finding the KPCA reconstruction of $\mathbf{x}$ is to optimize:

$$\arg \min_z ||\varphi(\mathbf{z}) - \varphi(\mathbf{x})||^2 \text{ s.t. } \varphi(\mathbf{z}) \in \mathcal{PS} . \tag{2}$$

However, since there might not exist $\mathbf{z} \in \mathcal{X}$ such that $\varphi(\mathbf{z}) \in \mathcal{PS}$, the above optimization problem needs to be relaxed. There is a common relaxation approach used by existing methods for computing the KPCA reconstruction of $\mathbf{x}$. This approach conceptually involves two steps:(i) finding $\mathbf{P}\varphi(\mathbf{x})$ which is the closest point to $\varphi(\mathbf{x})$ among all the points in the principal subspace, (ii) finding $\mathbf{z}$ such that $\varphi(\mathbf{z})$ is as close to $\mathbf{P}\varphi(\mathbf{x})$ as possible. This relaxation is depicted in Fig. 3a. If $L_2$ norm is used to measure the closeness between $\varphi(\mathbf{z})$ and $\mathbf{P}\varphi(\mathbf{x})$, the resulting KPCA reconstruction is $\arg \min_{\mathbf{z}} ||\varphi(\mathbf{z}) - \mathbf{P}\varphi(\mathbf{x})||_2^2$ .

This approach for KPCA reconstruction is not robust. For example, if $\mathbf{x}$ is corrupted with intra-sample outliers (e.g. occlusion), $\varphi(\mathbf{x})$ and $\mathbf{P}\varphi(\mathbf{x})$ will also be adversely affected. As a consequence, finding $\mathbf{z}$ that minimizes $||\varphi(\mathbf{z}) - \mathbf{P}\varphi(\mathbf{x})||_2^2$ does not always produce a "clean" version of $\mathbf{x}$. Furthermore, it is unclear how to incorporate robustness to the above formulation.

Here, we propose a novel relaxation of (2) that enables the incorporation of robustness. The KPCA reconstruction of $\mathbf{x}$ is taken as:

$$\arg \min_{\mathbf{z}} ||\varphi(\mathbf{x}) - \varphi(\mathbf{z})||_2^2 + C \underbrace{||\varphi(\mathbf{z}) - \mathbf{P}\varphi(\mathbf{z})||_2^2}_{E_{proj}(\mathbf{z})} . \tag{3}$$

---
**Algorithm 1** RKPCA for missing attribute values in training data
---
    **Input:** training data $\mathbf{D}$, number of iterations $m$, number of partitions $k$.
    **Initialize:** missing values by the means of known values.
    **for** $iter = 1$ **to** $m$ **do**
        Randomly divide $\mathbf{D}$ into $k$ equal partitions $\mathbf{D}_1, ..., \mathbf{D}_k$
        **for** $i = 1$ **to** $k$ **do**
            Train RKPCA using data $\mathbf{D} \setminus \mathbf{D}_i$
            Run RKPCA fitting for $\mathbf{D}_i$ with known missing attributes.
        **end for**
        Update missing values of $\mathbf{D}$
    **end for**
---

Intuitively, the above cost function requires the KPCA reconstruction of $\mathbf{x}$ is a point $\mathbf{z}$ that $\varphi(\mathbf{z})$ is close to both $\varphi(\mathbf{x})$ and the principal subspace. $C$ is a user-defined parameter that controls the relative importance of these two terms. This approach is depicted in Fig. 3b.

It is possible to generalize the above cost function further. The first term of Eq. 3 is not necessarily $||\varphi(\mathbf{x}) - \varphi(\mathbf{z})||_2^2$. In fact, for the sake of robustness, it is preferable that $||\varphi(\mathbf{x}) - \varphi(\mathbf{z})||_2^2$ is replaced by a robust function $E_0 : \mathcal{X} \times \mathcal{X} \to \Re$ for measuring similarity between $\mathbf{x}$ and $\mathbf{z}$. Furthermore, there is no reason why $E_0$ should be restricted to the metric of the feature space. In short, the KPCA reconstruction of $\mathbf{x}$ can be taken as:

$$\arg \min_{\mathbf{z}} E_0(\mathbf{x}, \mathbf{z}) + C E_{proj}(\mathbf{z}) \ . \tag{4}$$

By choosing appropriate forms for $E_0$, one can use KPCA to handle noise, missing data, and intra-sample outliers. We will show that in the following sections.

### 3.2 Dealing with missing data in testing samples

Assume the KPCA has been learned from complete and noise free data. Given a new image $\mathbf{x}$ with missing values, a logical function $E_0$ that does not depend on the the missing values could be: $E_0(\mathbf{x}, \mathbf{z}) = -\exp(-\gamma_2 ||\mathbf{W}(\mathbf{x} - \mathbf{z})||_2^2)$, where $\mathbf{W}$ is a diagonal matrix; the elements of its diagonal are 0 or 1 depending on whether the corresponding attributes of $\mathbf{x}$ have missing values or not.

### 3.3 Dealing with intra-sample outliers in testing samples

To handle intra-sample outliers, we could use a robust function for $E_0$. For instance: $E_0(\mathbf{x}, \mathbf{z}) = -\exp(-\gamma_2 \sum_{i=1}^{d} \rho(x_i - z_i, \sigma))$, where $\rho(\cdot, \cdot)$ is the Geman-McClure function, $\rho(y, \sigma) = \frac{y^2}{y^2 + \sigma^2}$, and $\sigma$ is a parameter of the function. This function is also used in [6] for Robust PCA.

### 3.4 Dealing with missing data and intra-sample outliers in training data

Previous sections have shown how to deal with outliers and missing data in the testing set (assuming KPCA has been learned from a clean training set). If we have missing data in the training samples [6], a simple approach is to iteratively alternate between estimating the missing values and updating the KPCA principal subspace until convergence. Algorithm 1 outlines the main steps of this approach. An algorithm for handling intra-sample outliers in training data could be constructed similarly.

Alternatively, a kernel matrix could be computed ignoring the missing values, that is, each $k_{ij} = \exp(-\gamma_2 ||\mathbf{W_i}\mathbf{W_j}(\mathbf{x_i} - \mathbf{x_j})||_2^2)$, where $\gamma_2 = \frac{1}{\text{trace}(\mathbf{W_i}\mathbf{W_j})}$. However, the positive definiteness of the resulting kernel matrix cannot be guaranteed.

### 3.5 Optimization

In general, the objective function in Eq. 4 is not concave, hence non-convex optimization techniques are required. In this section, we restrict our attention to the Gaussian kernel ($k(\mathbf{x}, \mathbf{y}) = \exp(-\gamma ||\mathbf{x} - \mathbf{y}||^2)$) that is the most widely used kernel. If $E_0$ takes the form of Sec.3.2, we need to

$$\text{maximize} \quad E(\mathbf{z}) = \underbrace{\exp(-\gamma_2 ||\mathbf{W}(\mathbf{x} - \mathbf{z})||^2)}_{E_1(\mathbf{z})} + C * \underbrace{r(\mathbf{z})^T \mathbf{M} r(\mathbf{z})}_{E_2(\mathbf{z})} \ , \tag{5}$$

where $r(\cdot)$, and $\mathbf{M}$ are defined in Eq.1. Note that optimizing this function is not harder than optimizing the objective function used by Mika *et al* [13]. Here, we also derive a fixed-point optimization algorithm. The necessary condition for a minimum has to satisfy the following equation: $\nabla_{\mathbf{z}} E(\mathbf{z}) = \nabla_{\mathbf{z}} E_1(\mathbf{z}) + \nabla_{\mathbf{z}} E_2(\mathbf{z}) = 0$ . The expression for the gradients are given by:

$$\nabla_{\mathbf{z}} E_1(\mathbf{z}) = -2\gamma_2 \underbrace{\exp(-\gamma_2 ||\mathbf{W}(\mathbf{x} - \mathbf{z})||^2) \mathbf{W}^2}_{\mathbf{W}_2} (\mathbf{z} - \mathbf{x}), \ \ \nabla_{\mathbf{z}} E_2(\mathbf{z}) = -4\gamma [(\mathbf{1}_n^T \mathbf{Q} \mathbf{1}_n)\mathbf{z} - \mathbf{D}\mathbf{Q}\mathbf{1}_n] ,$$

where $\mathbf{Q}$ is a matrix such that $q_{ij} = m_{ij} \exp(-\gamma ||\mathbf{z} - \mathbf{d}_i||^2 - \gamma ||\mathbf{z} - \mathbf{d}_j||^2)$. A fixed-point update is:

$$\mathbf{z} = [\underbrace{\frac{1}{2C}\frac{\gamma_2}{\gamma}\mathbf{W}_2 + (\mathbf{1}_n^T \mathbf{Q} \mathbf{1}_n)\mathbf{I}_n}_{\mathbf{W}_3}]^{-1} (\underbrace{\frac{1}{2C}\frac{\gamma_2}{\gamma}\mathbf{W}_2 \mathbf{x} + \mathbf{D}\mathbf{Q}\mathbf{1}_n}_{\mathbf{u}}) \tag{6}$$

The above equation is the update rule for $\mathbf{z}$ at every iteration of the algorithm. The algorithm stops when the difference between two successive $\mathbf{z}$'s is smaller than a threshold.

Note that $\mathbf{W}_3$ is a diagonal matrix with non-negative entries since $\mathbf{Q}$ is a positive semi-definite matrix. Therefore, $\mathbf{W}_3$ is not invertible only if there are some zero elements on the diagonal. This only happens if some elements of the diagonal of $\mathbf{W}$ are 0 and $\mathbf{1}_n^T \mathbf{Q} \mathbf{1}_n = 0$. It can be shown that $\mathbf{1}_n^T \mathbf{Q} \mathbf{1}_n = \sum_{i=1}^m (\mathbf{v}_i^T \varphi(\mathbf{z}))^2$, so $\mathbf{1}_n^T \mathbf{Q} \mathbf{1}_n = 0$ when $\varphi(\mathbf{z}) \perp \mathbf{v}_i, \forall i$. However, this rarely occurs in practice; moreover, if this happens we can restart the algorithm from a different initial point.

Consider the update rule given in Eq.6: $\mathbf{z} = \mathbf{W}_3^{-1} \mathbf{u}$. The diagonal matrix $\mathbf{W}_3^{-1}$ acts as a normalization factor of $\mathbf{u}$. Vector $\mathbf{u}$ is a weighted combination of two terms, the training data $\mathbf{D}$ and $\mathbf{x}$. Furthermore, each element of $\mathbf{x}$ is weighted differently by $\mathbf{W}_2$ which is proportional to $\mathbf{W}$. In the case of missing data (some entries in the diagonal of $\mathbf{W}$, and therefore $\mathbf{W}_2$, will be zero), missing components of $\mathbf{x}$ would not affect the computation of $\mathbf{u}$ and $\mathbf{z}$. Entries corresponding to the missing components of the resulting $\mathbf{z}$ will be pixel-weighted combinations of the training data. The contribution of $\mathbf{x}$ also depends on the ratio $\gamma_2/\gamma$, $C$, and the distance from the current $\mathbf{z}$ to $\mathbf{x}$. Similar to the observation of Mika *et al* [13], the second term of vector $\mathbf{u}$ pulls $\mathbf{z}$ towards a single Gaussian cluster. The attraction force generated by a training data point $\mathbf{d}_i$ reflects the correlation between $\varphi(\mathbf{z})$ and $\varphi(\mathbf{d}_i)$, the correlation between $\varphi(\mathbf{z})$ and eigenvectors $\mathbf{v}_j$'s, and the contributions of $\varphi(\mathbf{d}_i)$ to the eigenvectors. The forces from the training data, together with the attraction force by $\mathbf{x}$, draw $\mathbf{z}$ towards a Gaussian cluster that is close to $\mathbf{x}$.

One can derive a similar update rule for $\mathbf{z}$ if $E_0$ takes the form in Sec.3.3. $\mathbf{z} = [\frac{1}{2C}\frac{\gamma_2}{\gamma}\mathbf{W}_4 + (\mathbf{1}_n^T \mathbf{Q} \mathbf{1}_n)\mathbf{I}_n]^{-1}(\frac{1}{2C}\frac{\gamma_2}{\gamma}\mathbf{W}_4 \mathbf{x} + \mathbf{D}\mathbf{Q}\mathbf{1}_n)$, with $\mathbf{W}_4 = \exp(-\gamma_2 \sum_{i=1}^d \rho(x_i - z_i, \sigma))\mathbf{W}_5^2$, where $\mathbf{W}_5$ is a diagonal matrix; the $i^{th}$ entry of the diagonal is $\sigma/((z_i - x_i)^2 + \sigma^2)$. The parameter $\sigma$ is updated at every iteration as follows: $\sigma = 1.4826 \times \text{median}(\{|z_i - x_i|\}_{i=1}^d)$ [5].

## 4 Experiments

### 4.1 RKPCA for intra-sample outliers

In this section, we compare RKPCA with three approaches for handling intra-sample outliers: (i) Robust Linear PCA [6], (ii) Mika *et al*'s KPCA reconstruction [13], and (iii) Kwok & Tsang's KPCA reconstruction [10]. The experiments are done on the CMU Multi-PIE database [8].

The Multi-PIE database consists of facial images of 337 subjects taken under different illuminations, expressions and poses, at four different sessions. We only make use of the directly-illuminated frontal face images under five different expressions (smile, disgust, squint, surprise and scream), see Fig. 4. Our dataset contains 1100 images, 700 are randomly selected for training, 100 are used for validation, and the rest is reserved for testing. Note that no subject in the testing set appears in the training set. Each face is manually labeled with 68 landmarks, as shown in Fig. 4a. A shape-normalized face is generated for every face by warping it towards the mean shape using affine transformation. Fig. 4b shows an example of such a shape-normalized face. The mean shape is used as the face mask and the values inside the mask are vectorized.

To quantitatively compare different methods, we introduce synthetic occlusions of different sizes (20, 30, and 40 pixels) into the test images. For each occlusion size and test image pair, we generate

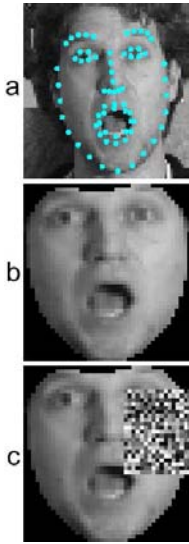

| | Occ.Sz | Region Type | Base Line | Mika et al | Kwok&Tsang | Robust PCA | Ours |
|---|---|---|---|---|---|---|---|
| **Energy 80%** | 20 | Whole face | 14.0±5.5 | 13.5±3.3 | 14.1±3.4 | 10.8±2.4 | **8.1±2.3** |
| | | Occ. Reg. | 71.5±5.5 | 22.6±7.9 | 17.3±6.6 | **13.3±5.5** | 16.1±6.1 |
| | | Non-occ Reg. | 0.0±0.0 | 11.3±2.3 | 13.2±2.9 | 10.1±2.2 | **6.0±1.7** |
| | 30 | Whole face | 27.7±10.2 | 17.5±4.8 | 16.6±4.6 | 12.2±3.2 | **10.9±4.2** |
| | | Occ. Reg. | 70.4±3.9 | 24.2±7.1 | 19.3±6.6 | **15.4±5.1** | 18.4±5.8 |
| | | Non-occ Reg. | 0.0±0.0 | 13.3±3.0 | 14.7±3.8 | 9.6±2.3 | **5.7±4.3** |
| | 40 | Whole face | 40.2±12.7 | 20.9±5.9 | 18.8±5.8 | 16.4±7.1 | **14.3±6.3** |
| | | Occ. Reg. | 70.6±3.6 | 25.7±7.2 | 21.1±7.1 | 20.1±8.0 | **19.8±6.3** |
| | | Non-occ Reg. | 0.0±0.0 | 15.2±4.2 | 16.1±5.3 | 9.4±2.3 | **8.8±8.1** |
| **Energy 95%** | 20 | Whole face | 14.2±5.3 | 12.6±3.1 | 13.8±3.2 | 9.1±2.3 | **7.0±2.1** |
| | | Occ. Reg. | 71.2±5.4 | 29.2±8.4 | **17.3±6.4** | 18.6±7.1 | 18.1±6.1 |
| | | Non-occ Reg. | 0.0±0.0 | 8.6±1.6 | 12.9±2.9 | 6.5±1.4 | **4.1±1.6** |
| | 30 | Whole face | 26.8±9.5 | 17.4±4.4 | 16.2±4.1 | 13.4±5.0 | **10.2±3.7** |
| | | Occ. Reg. | 70.9±4.4 | 30.0±7.6 | **19.5±6.5** | 23.8±7.8 | 21.0±6.3 |
| | | Non-occ Reg. | 0.0±0.0 | 10.1±1.9 | 14.1±3.2 | 6.3±1.4 | **3.1±1.7** |
| | 40 | Whole face | 40.0±11.9 | 22.0±5.9 | 18.9±6.0 | 22.7±11.7 | **14.3±5.8** |
| | | Occ. Reg. | 70.7±3.6 | 30.1±7.2 | **21.4±7.4** | 32.4±11.9 | 22.4±7.0 |
| | | Non-occ Reg. | 0.0±0.0 | 12.1±3.3 | 15.9±5.2 | 7.0±2.5 | **5.0±6.7** |

Figure 4: a) 68 landmarks, b) a shape-normalized face, c) synthetic occlusion.

Figure 5: Results of several methods on MPIE database. This shows the means and standard deviations of the absolute differences between reconstructed images and the ground-truths. The statistics are available for three types of face regions (whole face, occluded region, and non-occluded region), different occlusion sizes, and different energy settings. Our method consistently outperforms other methods for different occlusion sizes and energy levels.

a square occlusion window of that size, drawing the pixel values randomly from 0 to 255. A synthetic testing image is then created by pasting the occlusion window at a random position. Fig. 4c displays such an image with occlusion size of 20. For every synthetic testing image and each of the four algorithms, we compute the mean (at pixel level) of the absolute differences between the reconstructed image and the original test image without occlusion. We record these statistics for occluded region, non-occluded region and the whole face. The average statistics together with standard deviations are then calculated over the set of all testing images. These results are displayed in Fig. 5. We also experiment with several settings for the energy levels for PCA and KPCA. The energy level essentially means the number of components of PCA/KPCA subspace. In the interest of space, we only display results for two settings 80% and 95%. *Base Line* is the method that does nothing; the reconstructed images are exactly the same as the input testing images. As can be seen from Fig.5, our method consistently outperforms others for all energy levels and occlusion sizes (using the whole-face statistics). Notably, the performance of our method with the best parameter settings is also better than the performances of other methods with their best parameter settings.

The experimental results for Mika *et al*, Kwok & Tsang, Robust PCA [6] are generated using our own implementations. For Mika *et al*, Kwok & Tsang's methods, we use Gaussian kernels with $\gamma = 10^{-7}$. For our method, we use $E_0$ defined in Sec. 3.3. The kernel is Gaussian with $\gamma = 10^{-7}, \gamma_2 = 10^{-6}$, and $C = 0.1$. The parameters are tuned using validation data.

## 4.2 RKPCA for incomplete training data

To compare the ability to handle missing attributes in training data of our algorithm with other methods, we perform some experiments on the well known Oil Flow dataset [4]. This dataset is also used by Sanguinetti & Lawrence [16]. This dataset contains 3000 12-dimensional synthetically generated data points modeling the flow of a mixture of oil, water and gas in a transporting pipe-line.

We test our algorithm with different amount of missing data (from 5% to 50%) and repeat each experiment for 50 times. For each experiment, we randomly choose 100 data points and randomly remove some attribute values at some certain rate. We run Algorithm 1 to recover the values of the missing attributes, with $m = 25, k = 10, \gamma = 0.0375$ (same as [16]), $\gamma_2 = 0.0375, C = 10^7$. The squared difference between the reconstructed data and the original groundtruth data is measured, and the mean and standard deviation for 50 runs are calculated. Note that this experiment setting is exactly the same as the setting by [16].

Table 1: Reconstruction errors for 5 different methods and 10 probabilities of missing values for the Oil Flow dataset. Our method outperforms other methods for all levels of missing data.

| p(del) | 0.05 | 0.10 | 0.15 | 0.20 | 0.25 | 0.30 | 0.35 | 0.40 | 0.45 | 0.50 |
|---|---|---|---|---|---|---|---|---|---|---|
| mean | $13 \pm 4$ | $28 \pm 4$ | $43 \pm 7$ | $53 \pm 8$ | $70 \pm 9$ | $81 \pm 9$ | $97 \pm 9$ | $109 \pm 8$ | $124 \pm 7$ | $139 \pm 7$ |
| 1-NN | $5 \pm 3$ | $14 \pm 5$ | $30 \pm 10$ | $60 \pm 20$ | $90 \pm 20$ | NA | NA | NA | NA | NA |
| PPCA | $3.7 \pm .6$ | $9 \pm 2$ | $17 \pm 5$ | $25 \pm 9$ | $50 \pm 10$ | $90 \pm 30$ | $110 \pm 30$ | $110 \pm 20$ | $120 \pm 30$ | $140 \pm 30$ |
| PKPCA | $5 \pm 1$ | $12 \pm 3$ | $19 \pm 5$ | $24 \pm 6$ | $32 \pm 6$ | $40 \pm 7$ | $45 \pm 4$ | $61 \pm 8$ | $70 \pm 10$ | $100 \pm 20$ |
| Ours | $\mathbf{3.2 \pm 1.9}$ | $\mathbf{8 \pm 4}$ | $\mathbf{12 \pm 4}$ | $\mathbf{19 \pm 6}$ | $\mathbf{27 \pm 8}$ | $\mathbf{34 \pm 10}$ | $\mathbf{44 \pm 9}$ | $\mathbf{53 \pm 12}$ | $\mathbf{69 \pm 13}$ | $\mathbf{83 \pm 15}$ |

Experimental results are summarized in Tab. 1. The results of our method are shown in the last column. The results of other methods are copied verbatim from [16]. The *mean* method is a widely used heuristic where the missing value of an attribute of a data point is filled by the mean of known values of the same attribute of other data points. The *1-NN* method is another widely used heuristic in which the missing values are replaced by the values of the nearest point, where the pairwise distance is calculated using only the attributes with known values. *PPCA* is the probabilistic PCA method [11], and *PKPCA* is the method proposed by [16]. As can be seen from Tab. 1, our method outperforms other methods for all levels of missing data.

## 4.3   RKPCA for denoising

This section describes denoising experiments on the Multi-PIE database with Gaussian additive noise. For a fair evaluation, we only compare our algorithm with Mika *et al*'s, Kwok & Tsang's and Linear PCA. These are the methods that perform denoising based on subspaces and do not rely explicitly on the statistics of natural images. Quantitative evaluations show that the denoising ability of our algorithm is comparable with those of other methods.

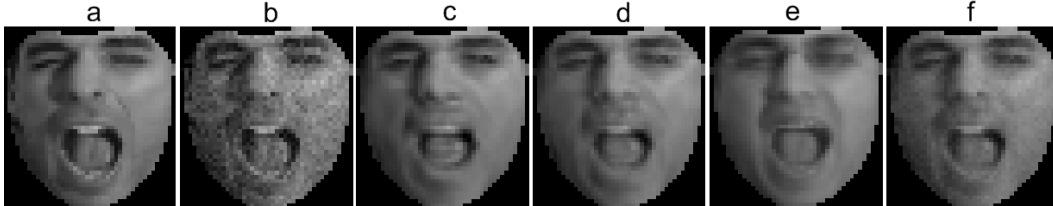

Figure 6: Example of denoised images. a) original image, b) corrupted by Gaussian noise, c) denoised using PCA, d) using Mika *et al*, e) using Kwok & Tsang method, f) result of our method.

The set of images used in these experiments is exactly the same as those in the occlusion experiments described in Sec. 4.1. For every testing image, we synthetically corrupt it with Gaussian additive noise with standard deviation of 0.04. An example for a pair of clean and corrupted images are shown in Fig. 6a and 6b. For every synthetic testing image, we compute the mean (at pixel level) of the absolute difference between the denoised image and the ground-truth. The results of different methods with different energy settings are summarized in Tab. 2. For these experiments, we use $E_0$ defined in Sec. 3.2 with $\mathbf{W}$ being the identity matrix. We use Gaussian kernel with $\gamma = \gamma_2 = 10^{-7}$, and $C = 1$. These parameters are tuned using validation data.

Table 2: Results of image denoising on the Multi-PIE database. *Base Line* is the method that does nothing. The best energy setting for all methods is 100%. Our method is better than the others.

| Energy | Base Line | Mika | Kwok& Tsang | PCA | Ours |
|---|---|---|---|---|---|
| 80% | 8.14±0.16 | 9.07±1.86 | 11.79±2.56 | 10.04±1.99 | **7.01±1.27** |
| 95% | 8.14±0.16 | 6.37±1.30 | 11.55±2.52 | 6.70±1.20 | **5.70±0.96** |
| 100% | 8.14±0.16 | 5.55±0.97 | 11.52±2.52 | 6.44±0.39 | **5.43±0.78** |

Tab. 2 and Fig. 6 show the performance of our method is comparable with others. In fact, the quantitative results show that our method is marginally better than Mika *et al*'s method and substantially better than the other two. In terms of visual appearance (Fig. 6c-f), the reconstruction image of our method preserves much more fine details than the others.

## 5  Conclusion

In this paper, we have proposed Robust Kernel PCA, a unified framework for handling noise, occlusion and missing data. Our method is based on a novel cost function for Kernel PCA reconstruction. The cost function requires the reconstructed data point to be close to the original data point as well as to the principal subspace. Notably, the distance function between the reconstructed data point and the original data point can take various forms. This distance function needs not to depend on the kernel function and can be evaluated easily. Therefore, the implicitness of the feature space is avoided and optimization is possible. Extensive experiments, in two well known data sets, show that our algorithm outperforms existing methods.

## Footnotes

[1]Bold uppercase letters denote matrices (e.g. $\mathbf{D}$), bold lowercase letters denote column vectors (e.g. $\mathbf{d}$). $\mathbf{d}_j$ represents the $j^{th}$ column of the matrix $\mathbf{D}$. $d_{ij}$ denotes the scalar in the row $i^{th}$ and column $j^{th}$ of the matrix $\mathbf{D}$ and the $i^{th}$ element of the column vector $\mathbf{d}_j$. Non-bold letters represent scalar variables. $\mathbf{1}_k \in \mathbb{R}^{k \times 1}$ is a column vector of ones. $\mathbf{I}_k \in \mathbb{R}^{k \times k}$ is the identity matrix.

## References

[1] Alzate, C. & Syukens, J.A. (2005) 'Robust Kernel Principal Component Analysis uisng Huber's Loss Function.' $24^{th}$ *Benelux Meeting on Systems and Control*.

[2] Bakir, G.H., Weston, J. & Schölkopf, B. (2004) 'Learning to Find Pre-Images.' in Thrun, S., Saul, L. & Schölkopf, B. (Eds) *Advances in Neural Information Processing Systems*.

[3] Berar, M., Desvignes, M., Bailly, G., Payan, Y. & Romaniuk, B. (2005) 'Missing Data Estimation using Polynomial Kernels.' *Proceedings of International Conference on Advances in Pattern Recognition*.

[4] Bishop, C.M., Svensén, M. & Williams, C.K.I. (1998) 'GTM: The Generative Topographic Mapping.' *Neural Computation*, **10**(1), 215–234.

[5] Black, M.J. & Anandan, P. (1996) 'The Robust Estimation of Multiple Motions: Parametric and Piecewise-smooth Flow Fields.' *Computer Vision and Image Understanding*, **63**(1), 75–104.

[6] de la Torre, F. & Black, M.J. (2003) 'A Framework for Robust Subspace Learning.' *International Journal of Computer Vision*, **54**(1–3), 117–142.

[7] Deng, X., Yuan, M. & Sudijanto, A. (2007) 'A Note on Robust Principal Component Analysis.' *Contemporary Mathematics*, **443**, 21–33.

[8] Gross, R., Matthews, I., Cohn, J., Kanade, T. & Baker, S. (2007) 'The CMU Multi-pose, Illumination, and Expression (Multi-PIE) Face Database.' Technical report, Carnegie Mellon University.TR-07-08.

[9] Jolliffe, I. (2002) *Principal Component Analysis*. 2 edn. Springer-Verlag, New York.

[10] Kwok, J.T.Y. & Tsang, I.W.H. (2004) 'The Pre-Image Problem in Kernel Methods.' *IEEE Transactions on Neural Networks*, **15**(6), 1517–1525.

[11] Lawrence, N.D. (2004) 'Gaussian Process Latent Variable Models for Visualization of High Dimensional Data.' in Thrun, S., Saul, L. & Schölkopf, B. (Eds) *Advances in Neural Information Processing Systems*.

[12] Lu, C., Zhang, T., Zhang, R. & Zhang, C. (2003) 'Adaptive Robust Kernel PCA Algorithm.' *Proceedings of IEEE International Conference on Acoustics, Speech, and Signal Processing*.

[13] Mika, S., Schölkopf, B., Smola, A., Müller, K.R., Scholz, M. & Rätsch, G. (1999) 'Kernel PCA and De-Noising in Feature Spaces.' *Advances in Neural Information Processing Systems*.

[14] Romdhani, S., Gong, S. & Psarrou, A. (1999) 'Multi-view Nonlinear Active Shape Model Using Kernel PCA.' *British Machine Vision Conference*, 483–492.

[15] Roweis, S. (1998) 'EM Algorithms for PCA and SPCA.' in Jordan, M., Kearns, M. & Solla, S. (Eds) *Advances in Neural Information Processing Systems 10*.

[16] Sanguinetti, G. & Lawrence, N.D. (2006) 'Missing Data in Kernel PCA.' *Proceedings of European Conference on Machine Learning*.

[17] Schölkopf, B., Mika, S., Smola, A., Rätsch, G. & Müller, K.R. (1998) 'Kernel PCA Pattern Reconstruction *via* Approximate Pre-Images.' *International Conference on Artificial Neural Networks*.

[18] Schölkopf, B. & Smola, A. (2002) *Learning with Kernels: Support Vector Machines, Regularization, Optimization, and beyond*. MIT Press, Cambridge, MA.

[19] Schölkopf, B., Smola, A. & Mller, K. (1998) 'Nonlinear Component Analysis as a Kernel Eigenvalue Problem.' *Neural Computation*, **10**, 1299–1319.

[20] Shawe-Taylor, J. & Cristianini, N. (2004) *Kernel Methods for Pattern Analysis*. Cambridge Uni. Press.

[21] Tipping, M. & Bishop, C.M. (1999) 'Probabilistic Principal Component Analysis.' *Journal of the Royal Statistical Society B*, **61**, 611–622.

[22] Wang, L., Pang, Y.W., Shen, D.Y. & Yu, N.H. (2007) 'An Iterative Algorithm for Robust Kernel Principal Component Analysis.' *Conference on Machine Learning and Cybernetics*.

